# Quality-Improved and Property-Preserved Polarimetric Imaging via Complementarily Fusing

**Chu Zhou**[1†]    **Yixing Liu**[2,3]    **Chao Xu**[4]    **Boxin Shi**[2,3*]

[1]National Institute of Informatics, Japan
[2]State Key Laboratory for Multimedia Information Processing, School of CS, Peking University, China
[3]National Engineering Research Center of Visual Technology, School of CS, Peking University, China
[4]National Key Laboratory of General Artificial Intelligence, School of IST, Peking University, China

zhou_chu@hotmail.com,
{luiginixy@stu., xuchao@cis., shiboxin@}pku.edu.cn

## Abstract

Polarimetric imaging is a challenging problem in the field of polarization-based vision, since setting a short exposure time reduces the signal-to-noise ratio, making the degree of polarization (DoP) and the angle of polarization (AoP) severely degenerated, while if setting a relatively long exposure time, the DoP and AoP would tend to be over-smoothed due to the frequently-occurring motion blur. This work proposes a polarimetric imaging framework that can produce clean and clear polarized snapshots by complementarily fusing a degraded pair of noisy and blurry ones. By adopting a neural network-based three-phase fusing scheme with specially-designed modules tailored to each phase, our framework can not only improve the image quality but also preserve the polarization properties. Experimental results show that our framework achieves state-of-the-art performance.

## 1    Introduction

Polarimetric imaging aims to obtain the degree of polarization (DoP) and the angle of polarization (AoP) of the scene to provide physical clues for downstream polarization-based vision applications (*e.g.*, reflection removal [16], shape from polarization [5], dehazing [31], *etc.*). In practice, the DoP and AoP cannot be captured directly, but are usually calculated from polarized snapshots[1] in an indirect manner. However, since a polarizer would block part of the light, selecting an appropriate exposure time could be challenging, making the captured polarized snapshots often degrade [15, 33]. As shown in Fig. 1 (left), setting a short exposure time would result in a very low signal-to-noise ratio (SNR), making the DoP and AoP severely degenerated; while if setting a relatively long exposure time to increase the SNR, motion blur is more likely to occur, making the DoP and AoP over-smoothed, as shown in Fig. 1 (middle). To deal with the above issues, several methods have been proposed to handle the low-light noise [10, 25, 15, 32] or motion blur [33] in the polarized images. Since these methods can work in a polarization-aware manner (*i.e.*, they explicitly take the preservation of polarization properties into consideration), they usually demonstrate higher performance compared with the corresponding methods designed for conventional images [3, 29, 14, 23]. However, due to the ill-posedness of the problems they face, the quality of their results is still limited.

Considering the fact that different types of degraded polarized snapshots would provide complementary knowledge, *i.e.*, the short-exposure noisy ones tend to be clear while the long-exposure blurry

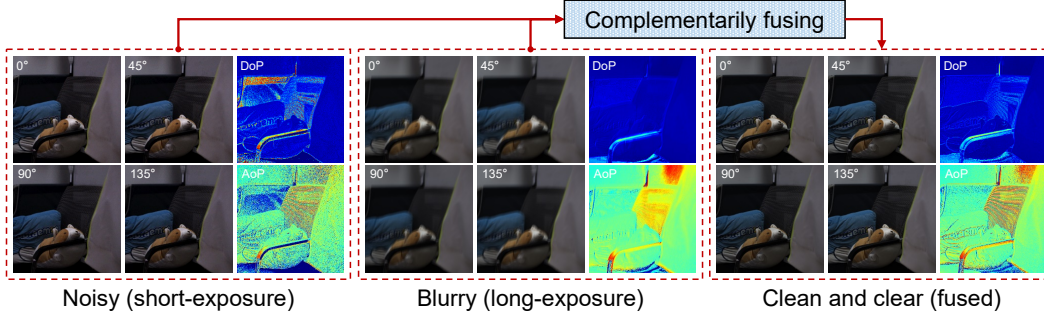

Complementarily fusing

Noisy (short-exposure)  Blurry (long-exposure)  Clean and clear (fused)

Figure 1: In polarimetric imaging, since a polarizer would block part of the light, setting a short exposure time would result in a low SNR, making the DoP and AoP severely degenerated (*left*); while if setting a relative long exposure time to increase the SNR, motion blur is more likely to occur, making the DoP and AoP over-smoothed (*middle*). Our framework can produce clean and clear results with high-quality DoP and AoP by complementarily fusing a degraded pair of noisy and blurry polarized snapshots (*right*).

ones tend to be clean, an intuitive strategy to improve the quality of polarimetric imaging could be complementarily fusing a degraded pair of noisy and blurry polarized snapshots. Such a strategy can not only achieve an effect similar to "boosting" (*i.e.*, combining multiple weak ones into a strong ones) to produce clean and clear results, but also bring the existing degraded polarized snapshots alive. However, current methods that can fuse noisy and blurry pairs [17, 2, 27, 28] are designed for conventional images, which are not suitable for polarimetric imaging since they cannot preserve the polarization properties, resulting in inaccurate DoP and AoP.

In this paper, we propose a quality-improved and property-preserved polarimetric imaging framework that can produce clean and clear polarized snapshots by complementarily fusing a degraded pair of noisy and blurry ones, as shown in Fig. 1 (right). Specifically, we design a neural network-based three-phase fusing scheme that can explicitly take the preservation of polarization properties into consideration. The first phase is *irradiance restoration*, aiming to restore the polarization-unrelated high-level irradiance information of the scene by recovering the total intensity of the light, where the color and structure cue fusion (CSCF) module is proposed to make full use of the color and structure cues encoded in the Stokes parameters. The second phase is *polarization reconstruction*, aiming to establish the physical correlation between the polarized images by reconstructing the DoP and AoP, where the coherence-aware aggregation (CAG) and coherence injection (CI) modules are proposed to optimize the values of the DoP and AoP in a Cartesian coordinate representation. The third phase is *artifact suppression*, aiming to suppress the artifacts lying in the details by performing refinement in the image domain. Unlike the fusing methods designed for conventional images [17, 2, 27, 28], our framework can fully utilize the complementary knowledge from the noisy and blurry pairs in a polarization-aware manner, by virtue of our three-phase fusing scheme. Different from the polarized image enhancement methods [32, 33] designed for enhancing a single noisy or blurry polarized snapshot, our framework can effectively explore the usage of different physical quantities to improve the overall performance, thanks to the specially-designed modules tailored to each phase. To summarize, this paper makes contributions by demonstrating:

- A quality-improved and property-preserved polarimetric imaging framework, for the first time applying a fusing strategy to polarimetric imaging.
- A neural network-based three-phase fusing scheme, fully utilizing the complementary knowledge from the noisy and blurry pairs in a polarization-aware manner.
- Specially-designed modules tailored to each phase, effectively exploring the usage of different physical quantities to improve the overall performance.

## 2   Related work

**Low-light enhancement and deblurring for polarized images.** There are many methods specially designed for enhancing polarized images, aiming to improve the quality of polarimetric imaging.

IPLNet [10] and ColorPolarNet [25] adopted residual dense blocks to build the backbone for dealing with multiple polarized low-light noisy images simultaneously. Li *et al*. [15] proposed a noise modeling method for realistic polarized low-light data synthesis along with a powerful vision Transformer-based network structure to reduce the noise. PLIE [32] designed a novel Stokes-domain low-light enhancement strategy and proposed a dual-branch network to reduce the artifacts lying in the DoP and AoP. PolDeblur [33] proposed a polarized image deblurring pipeline along with a two-stage network to remove the motion blur in a polarization-aware manner. However, they are not good at recovering details due to the ill-posedness of the problem they face.

**Image enhancement by fusing noisy and blurry pairs.** In comparison with the image enhancement methods that only take a single degenerated image as the input (*e.g.*, low-light enhancement [3, 29] or deblurring [14, 23] methods that focus on processing either a single low-light noisy image or a single blurry image), fusing noisy and blurry pairs could usually achieve higher performance and better generalization ability since additional information can be acquired. Early works are usually based on numerical optimization. Yuan *et al*. [26] adopted a residual deconvolution process along with a gain-controlled deconvolution process to reduce the overall ringing artifacts during fusing. Choi *et al*. [4] designed a novel camera system that could capture two blurry images and one noisy image in a single shot, and proposed a motion-based image merging algorithm to merge the captured images into a high-quality one. Son and Park [22] proposed a patches-based point spread function (PSF) estimation approach by extracting the structure information from the noisy image, along with a channel-dependent deblurring method to obtain the blur-free image. Son *et al*. [21] proposed a scheme to alternatively estimate the PSF and perform the deconvolution operation on the blurry image using the noisy image as a guiding signal. Gu *et al*. [7] proposed a method based on Gaussian mixture model to estimate the underlying intensity distribution of the noisy and blurry pairs first and then perform the pixel fusing. Recently, deep neural networks have also been adopted to handle this problem. LSD2 [17] and LSFNet [2] proposed to use convolutional neural networks to fuse the images in an end-to-end manner. SelfIR [27] proposed a self-supervised learning strategy to restore the clean and clear image contents. D2HNet [28] adopted a two-phase pipeline to further increase the visual quality. However, the above methods are designed to enhance the quality of a single input image, which would show inferior performance when handling multiple polarized images.

## 3 Method

### 3.1 Problem formulation and overall framework

As shown in Fig. 1, our goal is to reconstruct a clean and clear polarized snapshot with preservation of polarization properties (denoted as $\mathcal{I} = \mathbf{I}_{\alpha_{1,2,3,4}}$) from a degraded pair of noisy and blurry polarized snapshots (denoted as $\mathcal{L} = \mathbf{L}_{\alpha_{1,2,3,4}}$ and $\mathcal{B} = \mathbf{B}_{\alpha_{1,2,3,4}}$ respectively), where $\alpha_{1,2,3,4} = 0°, 45°, 90°, 135°$ stand for the polarizer angles of the polarized images in the polarized snapshot respectively. Once $\mathcal{I}$ becomes available, high-quality DoP $\mathbf{p}$ and AoP $\boldsymbol{\theta}$ could be calculated using

$$\mathbf{p} = \frac{\sqrt{\mathbf{S}_1^2 + \mathbf{S}_2^2}}{\mathbf{S}_0} \ \text{ and } \ \boldsymbol{\theta} = \frac{1}{2}\arctan(\frac{\mathbf{S}_2}{\mathbf{S}_1}), \tag{1}$$

where $\mathbf{S}_{0,1,2}$ are called the Stokes parameters [13][2], which can be computed as

$$\begin{cases} \mathbf{S}_0 = \frac{1}{2}(\mathbf{I}_{\alpha_1} + \mathbf{I}_{\alpha_2} + \mathbf{I}_{\alpha_3} + \mathbf{I}_{\alpha_4}) = \mathbf{I}_{\alpha_1} + \mathbf{I}_{\alpha_3} = \mathbf{I}_{\alpha_2} + \mathbf{I}_{\alpha_4} \\ \mathbf{S}_1 = \mathbf{I}_{\alpha_3} - \mathbf{I}_{\alpha_1} \\ \mathbf{S}_2 = \mathbf{I}_{\alpha_4} - \mathbf{I}_{\alpha_2} \end{cases}. \tag{2}$$

Here, we can see $\mathbf{S}_0$ describes the total intensity of the light, which is polarization-unrelated. In the following, we will use $\mathbf{S}_{0,1,2}^{\mathrm{L}}$ and $\mathbf{S}_{0,1,2}^{\mathrm{B}}$ to denote the Stokes parameters of $\mathcal{L}$ and $\mathcal{B}$ respectively.

The overall reconstruction process of our framework can be formulated as maximizing a posteriori of the output $\mathcal{I}$ conditioned on the inputs $\mathcal{L}$ and $\mathcal{B}$ along with the fusing function $f$ parameterized by $\Psi$:

$$\underset{\Psi}{\operatorname{argmax}} f(\mathcal{I}|\mathcal{L}, \mathcal{B}, \Psi). \tag{3}$$

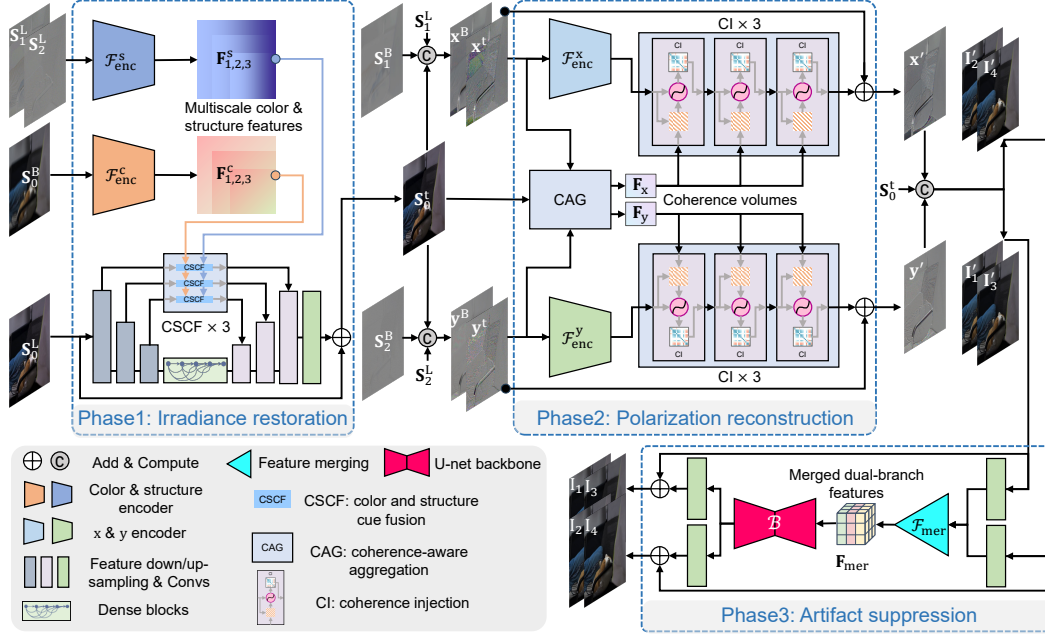

Figure 2: The workflow of our framework, consisting of three phases: irradiance restoration, polarization reconstruction, and artifact suppression.

To solve this maximum a posteriori estimation problem, we design a neural network-based three-phase fusing scheme to implement the fusing function $f$, as shown in Fig. 2. First, the irradiance restoration phase restore the polarization-unrelated high-level irradiance information of the scene, by enhancing $\mathbf{S}_0^L$ to obtain the coarse value of the total intensity of the light $\mathbf{S}_0^t$ under the guidance of the color and structure cues provided by $\mathbf{S}_0^B$ and $\mathbf{S}_{1,2}^B$ respectively. Then, we compute $(\mathbf{x}^t, \mathbf{y}^t)$, which are the coarse values of the DoP and AoP in a Cartesian coordinate representation, and feed them into the polarization reconstruction phase to obtain the corresponding enhanced values $(\mathbf{x}', \mathbf{y}')$ with the help of $(\mathbf{x}^B, \mathbf{y}^B)$ (the DoP and AoP of $\mathcal{B}$ in a Cartesian coordinate representation) along with the irradiance clues encoded in $\mathbf{S}_0^t$, aiming to establish the physical correlation between the polarized images. Finally, we compute the coarse values of the polarized images $\mathbf{I}'_{\alpha_{1,2,3,4}}$, and adopt an artifact suppression phase to obtain $\mathbf{I}_{\alpha_{1,2,3,4}}$ that make up the clean and clear polarized snapshot $\mathcal{I}$, by suppressing the artifacts in the image domain for increasing the quality of details.

### 3.2 Phase1: Irradiance restoration

This phase aims to restore the polarization-unrelated high-level irradiance information for providing further guidance. As shown in Fig. 1, since $\mathcal{L}$ would retain better contours than $\mathcal{B}$, we propose to learn the residual between $\mathbf{S}_0^L$ and $\mathbf{S}_0^t$ instead of the residual between $\mathbf{S}_0^B$ and $\mathbf{S}_0^t$. However, $\mathbf{S}_0^L$ usually suffers from color bias and noise, which would increase the difficulty of feature extraction, resulting in erroneous global tone and less salient local structure. Fortunately, despite that $\mathbf{S}_0^B$ would suffer from motion blur, it still contains undamaged color information due to the relatively high SNR of $\mathcal{B}$; besides, $\mathbf{S}_{1,2}^L$ could provide distinctive structure information since both of them describe the difference between two polarized images (see Eq. (2)), which would highlight the regions with high gradients. Therefore, we propose to effectively explore the usage of $\mathbf{S}_0^B$ and $\mathbf{S}_{1,2}^L$.

Specifically, we first explicitly adopt two modal-specific feature encoders $\mathcal{F}_{enc}^c$ and $\mathcal{F}_{enc}^s$ to extract the multiscale color and structure features $\mathbf{F}_{1,2,3}^c$ and $\mathbf{F}_{1,2,3}^s$ from $\mathbf{S}_0^B$ and $\mathbf{S}_{1,2}^B$ respectively for guidance. Then, we propose to use three color and structure cue fusion (CSCF) modules to apply the guidance provided by $\mathbf{F}_{1,2,3}^c$ and $\mathbf{F}_{1,2,3}^s$ to $\mathbf{F}_{1,2,3}^{in}$ in a successive manner, and output $\mathbf{F}_{1,2,3}^{out}$ for restoring $\mathbf{S}_0^t$. Here, $\mathbf{F}_{1,2,3}^{in}$ and $\mathbf{F}_{1,2,3}^{out}$ denote the multiscale input and output features of the CSCF modules respectively, which are extracted from and fed into the encoder and decoder part of a modified autoencoder architecture [9] where a dense block [11] is inserted into the coarsest layer.

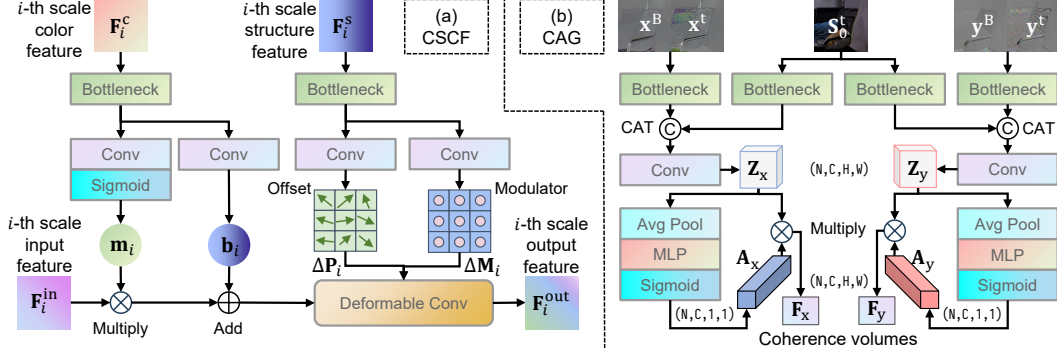

Figure 3: The details of the proposed CSCF (color and structure cue fusion) and CAG (coherence-aware aggregation) modules.

**CSCF: color and structure cue fusion.** The CSCF module aims to address the issues of erroneous global tone and less salient local structure in the feature space. Without losing generality, we describe how the $i$-th scale ($i = 1, 2, 3$) CSCF takes $\mathbf{F}_i^c$, $\mathbf{F}_i^s$, and $\mathbf{F}_i^{in}$ as the input and output $\mathbf{F}_i^{out}$, as shown in Fig. 3 (a). We first learn a multiplier $\mathbf{m}_i$ and a bias $\mathbf{b}_i$ from $\mathbf{F}_i^c$ by

$$\mathbf{m}_i = B_c(C_m(Sigmoid(\mathbf{F}_i^c))) \quad \text{and} \quad \mathbf{b}_i = B_c(C_b(\mathbf{F}_i^c)), \tag{4}$$

where $B_c$ denotes a bottleneck block [8] used for feature projection, $C_m$ and $C_b$ denote two different convolution layers. Then, we apply an affine transformation to $\mathbf{F}_i^{in}$ using $\mathbf{m}_i$ and $\mathbf{b}_i$, to adjust the color in the feature space for solving the issue of erroneous global tone by

$$\mathbf{F}_i^t = \mathbf{m}_i \odot \mathbf{F}_i^{in} + \mathbf{b}_i, \tag{5}$$

where $\mathbf{F}_i^t$ denote the transformed feature, $\odot$ denotes element-wise product operation. Finally, to solve the issue of less salient local structure, we apply a deformable convolution layer [34] $D$ to align the gradients and overcome the possible shifts caused by the exposure interval in the feature space by

$$\mathbf{F}_i^{out} = D(\mathbf{F}_i^t, \Delta\mathbf{P}_i, \Delta\mathbf{M}_i), \tag{6}$$

where $\Delta\mathbf{P}_i$ and $\Delta\mathbf{M}_i$ are the offsets of sampling points and the modulation scalars learned by

$$\Delta\mathbf{P}_i = B_s(C_P(\mathbf{F}_i^s)) \quad \text{and} \quad \Delta\mathbf{M}_i = B_s(C_M(\mathbf{F}_i^s)), \tag{7}$$

where $B_s$, $C_P$, and $C_M$ denotes another bottleneck block [8] and convolution layers respectively.

### 3.3 Phase2: Polarization reconstruction

This phase aims to establish the physical correlation between the polarized images by reconstructing the high-quality DoP and AoP. To achieve it, previous methods usually choose to repair the degenerated values in the image domain [10, 25, 15, 33] or Stokes domain [32] for an indirect reconstruction, since the degeneration patterns of the DoP and AoP could be more complicated than the polarized images or Stokes parameters due to their non-linearity (see Eq. (1)), which could increase the ill-posedness. In contrast, we propose to reconstruct the DoP and AoP in a Cartesian coordinate representation, which can not only relieve the ill-posedness since the non-linearity reduces, but also optimize the values of the DoP and AoP in a direct manner to prevent error accumulation.

Here, we explain what is the Cartesian coordinate representation of the DoP and AoP: as shown in Fig. 4, if we regard $\mathbf{p}$ and $2\theta$ in Eq. (1) as the magnitude and angle of a vector $\vec{\mathbf{S}}$ lying inside a unit circle, the Cartesian coordinate representation of $\vec{\mathbf{S}}$ could be written as $(\mathbf{x}, \mathbf{y})$, which satisfying

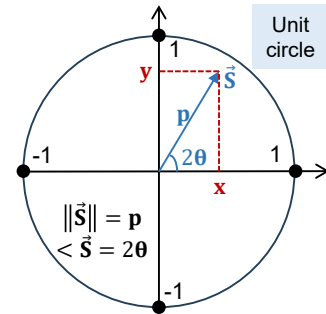

Figure 4: The Cartesian coordinate representation of the DoP and AoP.

$$\mathbf{x} = \frac{\mathbf{S}_1}{\mathbf{S}_0} \quad \text{and} \quad \mathbf{y} = \frac{\mathbf{S}_2}{\mathbf{S}_0}. \tag{8}$$

Specifically, we propose to learn the residual between $(\mathbf{x}^{\mathrm{t}}, \mathbf{y}^{\mathrm{t}})$ and $(\mathbf{x}', \mathbf{y}')$, with the help of $(\mathbf{x}^{\mathrm{B}}, \mathbf{y}^{\mathrm{B}})$ and $\mathbf{S}_0^{\mathrm{t}}$, where

$$\mathbf{x}^{\mathrm{t}} = \frac{\mathbf{S}_1^{\mathrm{L}}}{\mathbf{S}_0^{\mathrm{t}}} \ , \ \ \mathbf{y}^{\mathrm{t}} = \frac{\mathbf{S}_2^{\mathrm{L}}}{\mathbf{S}_0^{\mathrm{t}}} \ , \ \ \mathbf{x}^{\mathrm{B}} = \frac{\mathbf{S}_1^{\mathrm{B}}}{\mathbf{S}_0^{\mathrm{B}}} \ \ \text{and} \ \ \mathbf{y}^{\mathrm{B}} = \frac{\mathbf{S}_2^{\mathrm{B}}}{\mathbf{S}_0^{\mathrm{B}}}. \tag{9}$$

First, we propose to use a coherence-aware aggregation (CAG) module to estimate the coherence volumes $\mathbf{F}_{\mathrm{x}}$ and $\mathbf{F}_{\mathrm{y}}$ from $\mathbf{x}^{\mathrm{B,t}}$, $\mathbf{y}^{\mathrm{B,t}}$, and $\mathbf{S}_0^{\mathrm{t}}$. Then, we adopt two branches for reconstructing $\mathbf{x}'$ and $\mathbf{y}'$ respectively. The first branch contains a feature encoder $\mathcal{F}_{\mathrm{enc}}^{\mathrm{x}}$ and three cascaded coherence injection (CI) modules using $\mathbf{F}_{\mathrm{x}}$ for guidance. Similarly, the second branch contains a feature encoder $\mathcal{F}_{\mathrm{enc}}^{\mathrm{y}}$ and three cascaded CI modules using $\mathbf{F}_{\mathrm{y}}$ for guidance.

**CAG: coherence-aware aggregation.** The CAG module aims to aggregate the priors about the coherence between the polarization properties and the irradiance information. As shown in Fig. 3 (b), it contains two symmetrical parts for estimating $\mathbf{F}_{\mathrm{x}}$ and $\mathbf{F}_{\mathrm{y}}$ respectively. Here, we only describe how to estimate $\mathbf{F}_{\mathrm{x}}$ since the estimation process of $\mathbf{F}_{\mathrm{y}}$ could be similar. We first adopt two bottleneck blocks [8] to extract polarization features and irradiance features from $\mathbf{x}^{\mathrm{B,t}}$ and $\mathbf{S}_0^{\mathrm{t}}$ respectively. Then, we project the extracted features into the coherence features $\mathbf{Z}_{\mathrm{x}} \in \mathbb{R}^{N \times C \times H \times W}$ using a convolution layer, where $N, C, H, W$ represent the batch size, number of channels, height, and width respectively. After that, inspired by CBAM [24] that can make full use of the inter-channel relationship of features, we propose to learn an attention vector $\mathbf{A}_{\mathrm{x}} \in \mathbb{R}^{N \times C \times 1 \times 1}$ to recalibrate $\mathbf{Z}_{\mathrm{x}}$ for obtaining $\mathbf{F}_{\mathrm{x}} \in \mathbb{R}^{N \times C \times H \times W}$:

$$\mathbf{F}_{\mathrm{x}} = \mathbf{A}_{\mathrm{x}} \odot \mathbf{Z}_{\mathrm{x}} = P(MLP(Sigmoid(\mathbf{Z}_{\mathrm{x}}))) \odot \mathbf{Z}_{\mathrm{x}}, \tag{10}$$

where $P$ denotes the global average pooling operation and $MLP$ denotes a multi-layer perceptron.

**CI: coherence injection.** The CI module aims to inject the priors about coherence into the reconstruction of $\mathbf{x}^{\mathrm{t}}$ and $\mathbf{y}^{\mathrm{t}}$. To enlarge the receptive field and include long-range association, we choose to use Transformer modules [6] with cross-attention layers. Taking one of the CI module in the first branch (the branch for reconstructing $\mathbf{x}^{\mathrm{t}}$) as an example, denoting its input as $\mathbf{F}_{\mathrm{x}}^{\mathrm{in}}$ (from the previous CI module or $\mathcal{F}_{\mathrm{enc}}^{\mathrm{x}}$) and $\mathbf{F}_{\mathrm{x}}$ (from the CAG module), we let $\mathbf{F}_{\mathrm{x}}^{\mathrm{in}}$ to serve as the query vector and adopt convolution layers to learn the key vector and the value vector from both $\mathbf{F}_{\mathrm{x}}^{\mathrm{in}}$ and $\mathbf{F}_{\mathrm{x}}$.

### 3.4 Phase3: Artifact suppression

With $\mathbf{S}_0^{\mathrm{t}}$ and $(\mathbf{x}', \mathbf{y}')$ available, we could compute the coarse values of the polarized images $\mathbf{I}'_{\alpha_{1,2,3,4}}$. However, we should not output $\mathbf{I}'_{\alpha_{1,2,3,4}}$ directly since their quality is still not satisfying. This is because $\mathbf{S}_0^{\mathrm{t}}$ and $(\mathbf{x}', \mathbf{y}')$ are estimated from two different phases so that the irradiance-polarization consistency would break, bringing artifacts to the details. Therefore, we add an extra phase to refine $\mathbf{I}'_{\alpha_{1,2,3,4}}$ for increasing the quality of details by suppressing the artifacts in the image domain.

Specifically, we propose to learn the residual between $\mathbf{I}'_{\alpha_{1,2,3,4}}$ and $\mathbf{I}_{\alpha_{1,2,3,4}}$. According to Eq. (2), we choose to divide $\mathbf{I}'_{\alpha_{1,2,3,4}}$ into two groups ($\mathbf{I}'_{\alpha_{1,3}}$ and $\mathbf{I}'_{\alpha_{2,4}}$) first to ensure both of them contain the full irradiance information and each of them contains half of the polarization properties. Then, we use two convolution layers to extract the features of $\mathbf{I}'_{\alpha_{1,3}}$ and $\mathbf{I}'_{\alpha_{2,4}}$ respectively, and adopt a feature merger $\mathcal{F}_{\mathrm{mer}}$ to obtain the merged dual-group features $\mathbf{F}_{\mathrm{mer}}$. After that, a U-Net backbone [18] is used to perform pixel-wise multi-scale feature refinement on $\mathbf{F}_{\mathrm{mer}}$, and another two convolution layers are adopted for decoding the refined dual-group features into the residuals of each group.

## 4 Implementation details

**Loss function.** The total loss function can be written as

$$L = L_{\mathrm{s}} + L_{\mathrm{p}} + L_{\mathrm{r}}, \tag{11}$$

which consists of three terms to optimize the three phases respectively: irradiance loss $L_{\mathrm{s}}$, polarization loss $L_{\mathrm{p}}$, and refinement loss $L_{\mathrm{r}}$. The irradiance loss could be written as

$$L_{\mathrm{s}} = \lambda_{\mathrm{s}}^{\mathrm{a}} L_1(\mathbf{S}_0^{\mathrm{t}}, \mathbf{S}_0^{\mathrm{gt}}) + \lambda_{\mathrm{s}}^{\mathrm{b}} L_{\mathrm{perc}}(\mathbf{S}_0^{\mathrm{t}}, \mathbf{S}_0^{\mathrm{gt}}), \tag{12}$$

where $\lambda_{\mathrm{s}}^{\mathrm{a,b}}$ are set to be 10.0 and 0.05 respectively, $L_1$ and $L_{\mathrm{perc}}$ denote the $\ell_1$ loss and perceptual loss respectively, the subscript gt labels the ground truth throughout this paper. The perceptual loss

Table 1: Quantitative comparisons on synthetic data. The comparisons involve our framework, the state-of-the-art polarized image low-light enhancement method PLIE [32] and its improved version PLIE+, the only existing polarized image deblurring method PolDeblur [33] and its improved version PolDeblur+, and four learning-based image enhancement methods designed for conventional images that also fuse noisy and blurry pairs (LSD2 [17], LSFNet [2], SelfIR [27], and D2HNet [28]).

| | PSNR-p | SSIM-p | PSNR-$\theta$ | SSIM-$\theta$ | PSNR-$\mathbf{S}_0$ | SSIM-$\mathbf{S}_0$ |
|---|---|---|---|---|---|---|
| **Ours** | **29.23** | **0.797** | **16.96** | **0.382** | **39.05** | **0.982** |
| PLIE [32] | 27.91 | 0.790 | 15.92 | 0.371 | 38.95 | 0.978 |
| PLIE+ | 27.98 | 0.794 | 16.93 | 0.379 | 39.01 | 0.979 |
| PolDeblur [33] | 24.52 | 0.676 | 15.73 | 0.280 | 26.12 | 0.794 |
| PolDeblur+ | 25.31 | 0.758 | 16.75 | 0.374 | 39.04 | 0.981 |
| LSD2 [17] | 25.73 | 0.662 | 13.75 | 0.288 | 27.88 | 0.905 |
| LSFNet [2] | 25.56 | 0.693 | 15.90 | 0.282 | 26.76 | 0.826 |
| SelfIR [27] | 19.43 | 0.647 | 15.39 | 0.231 | 25.90 | 0.785 |
| D2HNet [28] | 24.45 | 0.671 | 15.63 | 0.264 | 25.25 | 0.803 |

$L_{\mathrm{perc}}$ is defined as

$$L_{\mathrm{perc}}(\mathbf{S}_0^{\mathrm{t}}, \mathbf{S}_0^{\mathrm{gt}}) = L_2(\phi_h(\mathbf{S}_0^{\mathrm{t}}), \phi_h(\mathbf{S}_0^{\mathrm{gt}})), \tag{13}$$

where $L_2$ denotes the $\ell_2$ loss, $\phi_h$ denotes the feature map from $h$-th layer of VGG-19 network [20] pretrained on ImageNet [19], and we use activations from $VGG_{3,3}$ convolution layer here. The polarization loss could be written as

$$L_{\mathrm{p}} = \lambda_{\mathrm{p}}^{\mathrm{a}}(L_1(\mathbf{x}', \mathbf{x}^{\mathrm{gt}}) + L_1(\mathbf{y}', \mathbf{y}^{\mathrm{gt}})) + \lambda_{\mathrm{p}}^{\mathrm{b}}(L_{\mathrm{tv}}(\mathbf{x}') + L_{\mathrm{tv}}(\mathbf{y}')) + \lambda_{\mathrm{p}}^{\mathrm{c}}L_{\mathrm{pol}}^1(\mathbf{x}', \mathbf{x}^{\mathrm{gt}}, \mathbf{y}', \mathbf{y}^{\mathrm{gt}}), \tag{14}$$

where $\lambda_{\mathrm{p}}^{\mathrm{a,b,c}}$ are set to be 1.0, 0.15, and 1.0 respectively, $L_{\mathrm{tv}}$ denotes the total variation loss, $L_{\mathrm{pol}}^1$ is a polarization-based regularization term to ensure the ratio between $\mathbf{x}'$ and $\mathbf{y}'$ defined as

$$L_{\mathrm{pol}}^1(\mathbf{x}', \mathbf{x}^{\mathrm{gt}}, \mathbf{y}', \mathbf{y}^{\mathrm{gt}}) = L_2(\mathbf{x}' \odot \mathbf{y}^{\mathrm{gt}}, \mathbf{y}' \odot \mathbf{x}^{\mathrm{gt}}). \tag{15}$$

The refinement loss could be written as

$$L_{\mathrm{r}} = \lambda_{\mathrm{r}}^{\mathrm{a}}L_1(\mathbf{I}_{\alpha_{1,2,3,4}}, \mathbf{I}_{\alpha_{1,2,3,4}}^{\mathrm{gt}}) + \lambda_{\mathrm{r}}^{\mathrm{b}}L_{\mathrm{pol}}^2(\mathbf{I}_{\alpha_{1,2,3,4}}), \tag{16}$$

where $\lambda_{\mathrm{r}}^{\mathrm{a,b}}$ are set to be 10.0 and 100.0 respectively, $L_{\mathrm{pol}}^2$ denotes another polarization-based regularization term defined as

$$L_{\mathrm{pol}}^2(\mathbf{I}_{\alpha_{1,2,3,4}}) = L_2(\mathbf{I}_{\alpha_1} + \mathbf{I}_{\alpha_3}, \mathbf{I}_{\alpha_2} + \mathbf{I}_{\alpha_4}), \tag{17}$$

which is similar to the one used in [30] for enforcing the preservation of polarization properties.

**Dataset preparation.** We propose to generate a synthetic dataset due to the fact that there is no public dataset for our settings. First, we choose the PLIE dataset [32] as our data source. It provides short-exposure polarized snapshots that suffer from low-light noise along with the corresponding high-quality reference snapshots captured by a Lucid Vision Phoenix polarization camera, which could serve as $\mathcal{L}$ and $\mathcal{I}$. Then, we adopt the approach proposed in [33] to generate the blurry polarized snapshots that suffer from motion blur, which could be served as $\mathcal{B}$. To generate more severe motion blur for increasing the diversity, we add impulsive variation [1] to the motion trajectories. The images are resized and randomly cropped to $256 \times 256$ ($512 \times 512$) pixels in the training (test) set. The training (test) set contains 7500 (300) different images in total.

**Training strategy.** Our framework is implemented using PyTorch with 2 NVIDIA 2080Ti GPUs, and a two-stage training strategy is applied. First, to ensure a stable initialization of the training process, we train the irradiance restoration phase and the polarization reconstruction phase independently for 300 epochs with learning rates of 0.01 and 0.0001 respectively. Then, we train the entire network for 100 epochs with learning rate of 0.0001, and in this training stage we multiply the loss terms $L_{\mathrm{s,p,r}}$ with 5.0, 10.0, and 10.0 respectively. For optimization, we use Adam optimizer [12] with $\beta_1 = 0.5$, $\beta_2 = 0.999$.

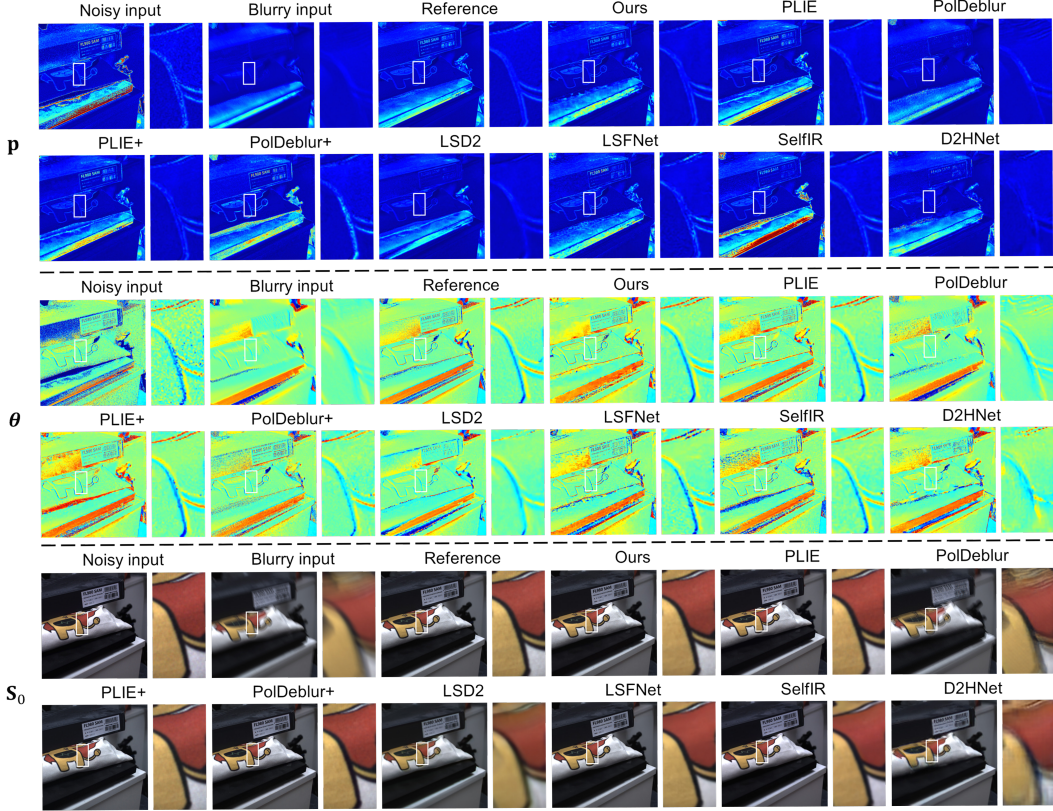

Figure 5: Qualitative comparisons on synthetic data. See the caption of Tab. 1 for explanation. We visualize the DoP $\mathbf{p}$ and AoP $\boldsymbol{\theta}$ using color maps after normalizing and averaging the RGB channels (as done in other methods designed for polarized images [10, 25, 15, 32, 33]) throughout this paper.

# 5 Experiments

## 5.1 Evaluation on synthetic data

First, we compare our framework with the state-of-the-art polarized image low-light enhancement method PLIE [32] and the only existing polarized image deblurring method PolDeblur [33]. Besides, we also compared with PLIE+ and PolDeblur+ (the improved versions of PLIE [32] and PolDeblur [33], where slight modifications are made to allow them to accept noisy and blurry pairs as the input) to ensure a fair comparison. In addition, four learning-based image enhancement methods designed for conventional images that also fuse noisy and blurry pairs (LSD2 [17], LSFNet [2], SelfIR [27], and D2HNet [28]) are also compared to ensure a comprehensive evaluation. Note that all compared methods are retrained on our dataset. As other methods designed for polarized images [10, 25, 15, 32, 33] do, we not only evaluate the quality of $\mathbf{p}$ and $\boldsymbol{\theta}$, but also the quality of $\mathbf{S}_0$.

To evaluate the results quantitatively, we adopt two frequently-used metrics including PSNR and SSIM. Results are shown in Tab. 1, where our framework consistently outperforms the compared methods on all metrics. These results could demonstrate three points:

(1) *Complementarily fusing can improve the quality of polarimetric imaging*, which could be deduced from the fact that PLIE+ and PolDeblur+ (which can accept a degraded pair of noisy and blurry polarized snapshots as the input) achieve better performance compared with PLIE [32] and PolDeblur [33] (which can only accept a single degraded polarized snapshot as the input) respectively.

(2) *Our framework can utilize the complementary knowledge in a more effective manner*, since our framework still outperforms PLIE+ and PolDeblur+ despite that they share the same kind of input as our framework.

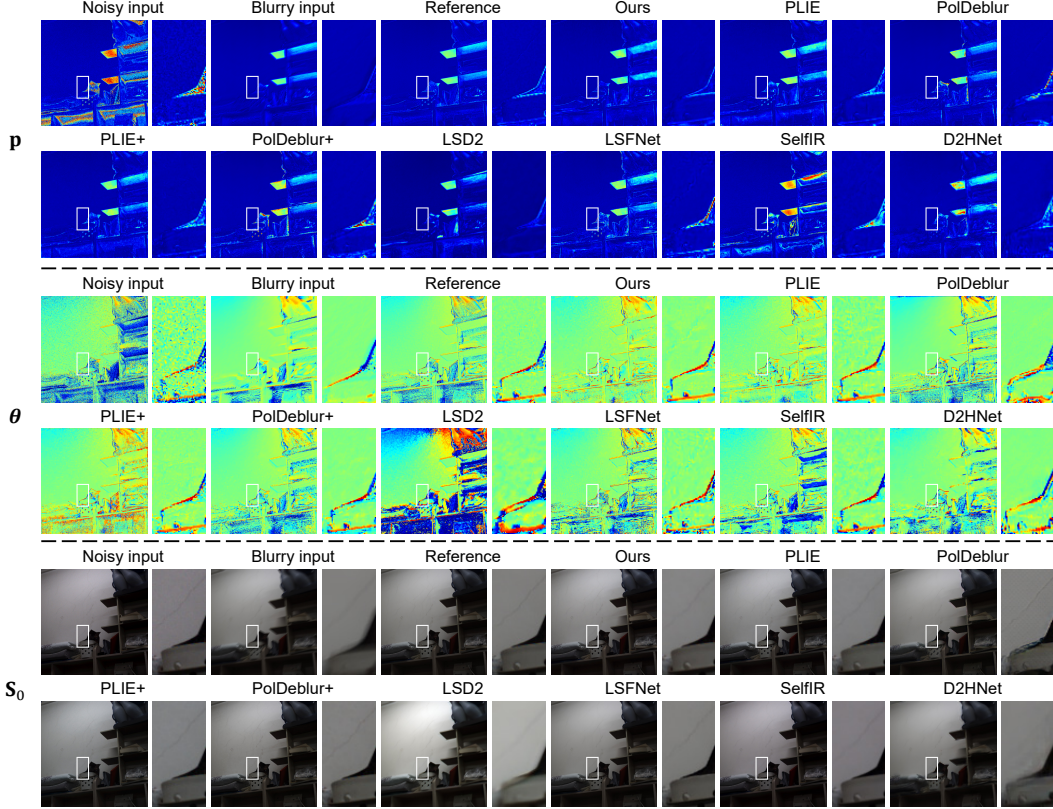

Figure 6: Qualitative comparisons on real data. See the caption of Fig. 5 for explanation.

    (3) *Designing a fusing framework tailored to the polarized images is necessary*, since the performance of both LSD2 [17], LSFNet [2], SelfIR [27], and D2HNet [28] is inferior in the task of polarimetric imaging.

Visual quality comparisons are shown in Fig. 5[3]. As for $\mathbf{p}$ and $\boldsymbol{\theta}$, our framework can produce clean and clear edges, since it can make full use of the complementary knowledge in a polarization-aware manner, while the compared methods suffer from noisy or blurry artifacts. As for $\mathbf{S}_0$, our results resemble the reference more closely with less color and structure distortion.

## 5.2   Evaluation on real data and downstream application

To evaluate on real data, we capture several pairs of noisy and blurry polarized snapshots from various scenes using a Lucid Vision Phoenix polarization camera. Qualitative results are shown in Fig. 6[4], from which we can see our framework can produce high-quality details.

Besides, in Fig. 7 we show that complementarily fusing can be beneficial to downstream polarization-based vision application such as reflection removal. Here, we feed a short-exposure noisy, a long-exposure blurry, and the fused polarized snapshots into a reflection removal network (RSP [16]) respectively. From the results we can see the reflection-removed image with the fusing process of our framework contains more detailed textures and less reflection contamination.

## 5.3   Ablation study

To verify the validity of each design choice, we conduct a series of ablation studies and show comparisons in Tab. 2. First, we show the effectiveness of the first phase by substituting it with

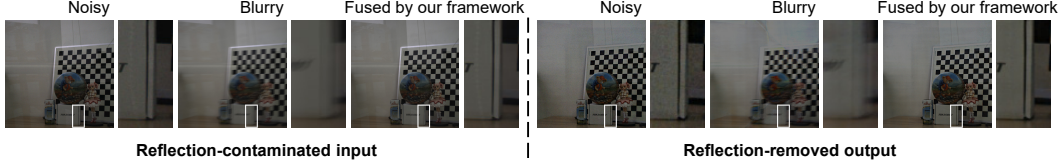

Figure 7: Results of reflection removal (using RSP [16]). Please zoom-in for better details.

Table 2: Quantitative evaluation results of ablation study.

|  | PSNR-p | SSIM-p | PSNR-$\theta$ | SSIM-$\theta$ | PSNR-$S_0$ | SSIM-$S_0$ |
|---|---|---|---|---|---|---|
| LSD2 [17] as Phase1 | 29.12 | 0.794 | 16.91 | 0.376 | 38.98 | 0.980 |
| W/o CSCF | 29.21 | 0.796 | 16.93 | 0.378 | 38.94 | 0.978 |
| W/o CAG | 29.07 | 0.793 | 16.87 | 0.371 | 38.97 | 0.980 |
| W/o CI | 28.92 | 0.788 | 16.90 | 0.371 | 38.90 | 0.976 |
| W/o Cartesian | 28.04 | 0.785 | 16.58 | 0.363 | 38.84 | 0.971 |
| W/o refinement | 29.02 | 0.791 | 16.84 | 0.367 | 38.89 | 0.976 |
| **Complete framework** | **29.23** | **0.797** | **16.96** | **0.382** | **39.05** | **0.982** |

a learning-based image enhancement methods designed for conventional images (LSD2 [17] as Phase1). The reason why we choose LSD2 [17] is that it outperforms other similar methods (LSFNet [2], SelfIR [27], and D2HNet [28]) in restoring $S_0$ (see Tab. 1). We can see that the performance becomes inferior since our first phase can make full use of the color and structure cues encoded in the Stokes parameters while LSD2 [17] cannot. Then, we show the effectiveness of the proposed CSCF modules in the first phase (W/o CSCF) by substituting them with vanilla convolution layers. We can see that the performance degenerates since the CSCF modules are more suitable for addressing the issues of erroneous global tone and less salient local structure in the feature space. Similarly, we also show the effectiveness of the proposed CAG module and CI modules in the second phase (W/o CAG and W/o CI). We can see that the CAG module can facilitate the establishment of the physical correlation between the polarized images, and the CI modules can offer better optimization to the values of the DoP and AoP. Besides, we demonstrate the advantage of reconstructing the DoP and AoP in a Cartesian coordinate representation (W/o Cartesian) by directly reconstructing their values. We can see that the performance degraded severely due to the non-linearity. Finally, we verify the necessity of the third phase used for refinement (W/o refinement) by removing it. These results show our complete framework achieves the first performance with the proposed specific designs.

## 6 Conclusion

We propose a quality-improved and property-preserved polarimetric imaging framework by complementarily fusing a degraded pair of noisy and blurry polarized snapshots. By adopting a neural network-based three-phase fusing scheme consisting of irradiance restoration, polarization reconstruction, and artifact suppression, with specially-designed modules tailored to each phase, our framework can produce clean and clear polarized snapshots with high-quality DoP and AoP.

**Limitations.** Since our framework is designed for reconstructing a single high-quality polarized snapshot from a degraded pair of noisy and blurry polarized snapshots, it cannot reconstruct a polarized video. Besides, it cannot be used to fuse conventional RGB images since our first phase requires the Stokes parameters as part of the input, which are not available in such a setting.

## Acknowledgments and Disclosure of Funding

This work was supported in part by National Natural Science Foundation of China under Grant No. 62136001, 62088102, 62276007 and JST-Mirai Program Grant Number JPMJM123G1.

## Footnotes

† Most of this work was done as a PhD student at Peking University.

[1]A polarized snapshot consists of four polarized images with different polarizer angles ($0°, 45°, 90°, 135°$), which can be captured using a polarization camera in a single shot or using a polarizer in multiple shots.

[2]More information about the Stokes parameters could be found in the supplementary material.

[3]Additional results can be found in the supplementary material.

[4]Please see the supplementary material for additional results.

# References

[1] Giacomo Boracchi and Alessandro Foi. Modeling the performance of image restoration from motion blur. *IEEE Transactions on Image Processing*, 21(8):3502–3517, 2012.

[2] Meng Chang, Huajun Feng, Zhihai Xu, and Qi Li. Low-light image restoration with short-and long-exposure raw pairs. *IEEE Transactions on Multimedia*, 24:702–714, 2021.

[3] Chen Chen, Qifeng Chen, Jia Xu, and Vladlen Koltun. Learning to see in the dark. In *Proc. of Computer Vision and Pattern Recognition*, 2018.

[4] Byeong-Doo Choi, Seung-Won Jung, and Sung-Jea Ko. Motion-blur-free camera system splitting exposure time. *IEEE Transactions on Consumer Electronics*, 54(3):981–986, 2008.

[5] Valentin Deschaintre, Yiming Lin, and Abhijeet Ghosh. Deep polarization imaging for 3D shape and SVBRDF acquisition. In *Proc. of Computer Vision and Pattern Recognition*, 2021.

[6] Alexey Dosovitskiy, Lucas Beyer, Alexander Kolesnikov, Dirk Weissenborn, Xiaohua Zhai, Thomas Unterthiner, Mostafa Dehghani, Matthias Minderer, Georg Heigold, Sylvain Gelly, et al. An image is worth $16 \times 16$ words: Transformers for image recognition at scale. *arXiv preprint arXiv:2010.11929*, 2020.

[7] Chunzhi Gu, Xuequan Lu, Ying He, and Chao Zhang. Blur removal via blurred-noisy image pair. *IEEE Transactions on Image Processing*, 30:345–359, 2020.

[8] Kaiming He, Xiangyu Zhang, Shaoqing Ren, and Jian Sun. Deep residual learning for image recognition. In *Proc. of Computer Vision and Pattern Recognition*, 2016.

[9] Geoffrey E Hinton and Ruslan R Salakhutdinov. Reducing the dimensionality of data with neural networks. *Science*, 313(5786):504–507, 2006.

[10] Haofeng Hu, Yang Lin, Xiaobo Li, Pengfei Qi, and Tiegen Liu. IPLNet: A neural network for intensity-polarization imaging in low light. *Optics Letters*, 45(22):6162–6165, 2020.

[11] Gao Huang, Zhuang Liu, Laurens Van Der Maaten, and Kilian Q Weinberger. Densely connected convolutional networks. In *Proc. of Computer Vision and Pattern Recognition*, 2017.

[12] Diederik P Kingma and Jimmy Ba. ADAM: A method for stochastic optimization, 2014.

[13] GP Können. *Polarized light in nature*. CUP Archive, 1985.

[14] Orest Kupyn, Volodymyr Budzan, Mykola Mykhailych, Dmytro Mishkin, and Jiří Matas. DeblurGAN: Blind motion deblurring using conditional adversarial networks. In *Proc. of Computer Vision and Pattern Recognition*, pages 8183–8192, 2018.

[15] Zhuoxiao Li, Haiyang Jiang, Mingdeng Cao, and Yinqiang Zheng. Polarized color image denoising. In *Proc. of Computer Vision and Pattern Recognition*, pages 9873–9882, 2023.

[16] Youwei Lyu, Zhaopeng Cui, Si Li, Marc Pollefeys, and Boxin Shi. Reflection separation using a pair of unpolarized and polarized images. In *Proc. of Advances in Neural Information Processing Systems*, 2019.

[17] Janne Mustaniemi, Juho Kannala, Jiri Matas, Simo Särkkä, and Janne Heikkilä. LSD2 - joint denoising and deblurring of short and long exposure images with CNNs. In *Proc. of British Machine Vision*, 2020.

[18] Olaf Ronneberger, Philipp Fischer, and Thomas Brox. U-Net: Convolutional networks for biomedical image segmentation. In *Proc. of International Conference on Medical Image Computing and Computer Assisted Intervention*, pages 234–241, 2015.

[19] Olga Russakovsky, Jia Deng, Hao Su, Jonathan Krause, Sanjeev Satheesh, Sean Ma, Zhiheng Huang, Andrej Karpathy, Aditya Khosla, Michael Bernstein, Alexander C Berg, and Li Fei-Fei. ImageNet large scale visual recognition challenge. *International Journal of Computer Vision*, 115(3):211–252, 2015.

[20] Karen Simonyan and Andrew Zisserman. Very deep convolutional networks for large-scale image recognition. *arXiv preprint arXiv:1409.1556*, 2014.

[21] Chang-Hwan Son, Hyunseung Choo, and Hyung-Min Park. Image-pair-based deblurring with spatially varying norms and noisy image updating. *Journal of Visual Communication and Image Representation*, 24(8):1303–1315, 2013.

[22] Chang-Hwan Son and Hyung-Min Park. A pair of noisy/blurry patches-based PSF estimation and channel-dependent deblurring. *IEEE Transactions on Consumer Electronics*, 57(4):1791–1799, 2011.

[23] Xin Tao, Hongyun Gao, Xiaoyong Shen, Jue Wang, and Jiaya Jia. Scale-recurrent network for deep image deblurring. In *Proc. of Computer Vision and Pattern Recognition*, pages 8174–8182, 2018.

[24] Sanghyun Woo, Jongchan Park, Joon-Young Lee, and In So Kweon. CBAM: Convolutional block attention module. In *Proc. of European Conference on Computer Vision*, pages 3–19, 2018.

[25] Xiuyu Xu, Minjie Wan, Junyu Ge, Hao Chen, Xingcheng Zhu, Xiaojie Zhang, Qian Chen, and Guohua Gu. ColorPolarNet: Residual dense network-based chromatic intensity-polarization imaging in low-light environment. *IEEE Transactions on Instrumentation and Measurement*, 71:1–10, 2022.

[26] Lu Yuan, Jian Sun, Long Quan, and Heung-Yeung Shum. Image deblurring with blurred/noisy image pairs. *ACM Transactions on Graphics (Proc. of ACM SIGGRAPH)*, 26(3):1–es, 2007.

[27] Zhilu Zhang, RongJian Xu, Ming Liu, Zifei Yan, and Wangmeng Zuo. Self-supervised image restoration with blurry and noisy pairs. *Advances in Neural Information Processing Systems*, 35:29179–29191, 2022.

[28] Yuzhi Zhao, Yongzhe Xu, Qiong Yan, Dingdong Yang, Xuehui Wang, and Lai-Man Po. D2HNet: Joint denoising and deblurring with hierarchical network for robust night image restoration. In *Proc. of European Conference on Computer Vision*, pages 91–110, 2022.

[29] Chuanjun Zheng, Daming Shi, and Wentian Shi. Adaptive unfolding total variation network for low-light image enhancement. In *Proc. of International Conference on Computer Vision*, 2021.

[30] Chu Zhou, Yufei Han, Minggui Teng, Jin Han, Si Li, Chao Xu, and Boxin Shi. Polarization guided HDR reconstruction via pixel-wise depolarization. *IEEE Transactions on Image Processing*, 32:1774–1787, 2023.

[31] Chu Zhou, Minggui Teng, Yufei Han, Chao Xu, and Boxin Shi. Learning to dehaze with polarization. In *Proc. of Advances in Neural Information Processing Systems*, 2021.

[32] Chu Zhou, Minggui Teng, Youwei Lyu, Si Li, Chao Xu, and Boxin Shi. Polarization-aware low-light image enhancement. In *Proc. of the AAAI Conference on Artificial Intelligence*, pages 3742–3750, 2023.

[33] Chu Zhou, Minggui Teng, Xinyu Zhou, Chao Xu, and Boxin Sh. Learning to deblur polarized images. *arXiv preprint arXiv:2402.18134*, 2024.

[34] Xizhou Zhu, Han Hu, Stephen Lin, and Jifeng Dai. Deformable ConvNets v2: More deformable, better results. In *Proc. of Computer Vision and Pattern Recognition*, pages 9308–9316, 2019.

# A Appendix

Please see the supplemental material for the things mentioned in Footnote 2, 3, and 4.

